# Causal inference in sensorimotor integration

**Konrad P. Körding**
Department of Physiology and PM&R
Northwestern University
Chicago, IL 60611
konrad@koerding.com

**Joshua B. Tenenbaum**
Massachusetts Institute of Technology
Cambridge, MA 02139
jbt@mit.edu

## Abstract

Many recent studies analyze how data from different modalities can be combined. Often this is modeled as a system that optimally combines several sources of information about the same variable. However, it has long been realized that this information combining depends on the interpretation of the data. Two cues that are perceived by different modalities can have different causal relationships: (1) They can both have the same cause, in this case we should fully integrate both cues into a joint estimate. (2) They can have distinct causes, in which case information should be processed independently. In many cases we will not know if there is one joint cause or two independent causes that are responsible for the cues. Here we model this situation as a Bayesian estimation problem. We are thus able to explain some experiments on visual auditory cue combination as well as some experiments on visual proprioceptive cue integration. Our analysis shows that the problem solved by people when they combine cues to produce a movement is much more complicated than is usually assumed, because they need to infer the causal structure that is underlying their sensory experience.

## 1 Introduction

Our nervous system is constantly integrating information from many different sources into a unified percept. When we interact with objects for example we see them and feel them and often enough we can also hear them. All these pieces of information need to be combined into a joint percept.

Traditionally, cue combination is formalized as a simple weighted combination of estimates coming from each modality (Fig 1A). According to this view the nervous system acquires these weights through some learning process [1]. Recently many experiments have shown that various manipulations, such as degrading the quality of the feedback from one modality, can vary the weights. Recently, these experiments have been phrased in a Bayesian framework, assuming that all the cues are about one given variable. Research often focuses on exploring in which coordinate system the problem is being solved [2, 3] and how much weight is given to each variable as a function of the uncertainty in each modality and the prior[4, 5, 6, 7, 8]. Throughout this paper we consider cue combination to estimate a position. Cue combination may, however, be equally important when estimating many other variables such as the nature of material, the weight of an object or the relevant aspects of a social situation.

These studies focus on the way information is combined and assume that is known that there is just one cause for the cues. However, in many cases people can not be certain of the causal structure. If two cues share a common cause (as in Fig 1B) they should clearly be combined. In general, however, there may either be one common cause – or two separate causes(Fig 1C). In such cases people can not know which of the two models to use and have to estimate the causal structure of the problem along with the parameter values. The issue of causal inference has long been an exciting question in

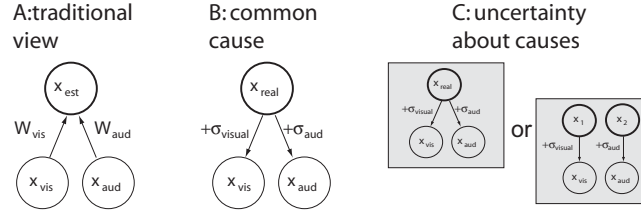

Figure 1: Different causal structures of two cues. Bold circles indicate the variables the subjects are interested in. A) The traditional view is sketched where the estimate is a weighted combination of the estimates of each modality. B) One cause can be responsible for both cues. In this case cues should be combined to infer about the single cause. C) In many cases people will be unable to know if one common cause or two independent causes are responsible for the cues. In that case people will have to estimate which causal structure is present from the properties of their sensory stimuli.

the psychological community [9, 10, 11, 12]. Here we derive a rigorous model of causal inference in the context of psychophysical experiments.

## 2   Cue combination: one common cause

A large number of recent studies have interpreted the results from cue combination studies in a Bayesian framework[13]. We discuss the case of visuoauditory integration as the statistical relations are identical in other cue combination cases. A statistical generative model for the data is formulated (see figure 1B). It is acknowledged that if a signal is coming from a specific position the signal received by the nervous system in each modality will be noisy. If the real position of a stimulus is $x_{real}$ then the nervous system will not be able to directly know this variable but the visual modality will obtain a noisy estimate thereof $x_{vis}$. Typically it is assumed that in the process that the visually perceived position is a noisy version of the real position $x_{vis} = x_{real} + noise$. A statistical treatment thus results in $p(x_{vis}|x_{real}) = N(x_{real} - x_{vis}, \sigma_{vis})$ where $\sigma_{vis}$ is the variance introduced by the visual modality and $N(\mu, \sigma)$ stands for a Gaussian distribution with mean $\mu$ and standard deviation $\sigma$. If two cues are available, for example vision and audition then it is assumed that both cues $x_{vis}$ and $x_{aud}$ provide noisy observations of the relevant variable $x_{real}$. Using the assumption that each modality provides an independent measurement of $x_{real}$ Bayes rule yields:

$$
\begin{align}
p(x_{real}|x_{vis}, x_{aud}) &\propto p(x_{real})p(x_{vis}, x_{aud}|x_{real}) \tag{1} \\
&= p(x_{real})p(x_{vis}|x_{real})p(x_{aud}|x_{real}) \tag{2}
\end{align}
$$

The estimate that minimizes the mean squared error is then:

$$
\hat{x} = \gamma x_{vis} + (1 - \gamma)x_{aud} \tag{3}
$$

where $\gamma = \sigma_{aud}^2/(\sigma_{aud}^2 + \sigma_{vis}^2)$. The optimal solution is thus a weighing of the estimates from both modalities but the weighing is a function of the variances. Given the variances of the cues, it is possible to predict the weighing people should optimally use. Over the last couple of years various studies have described this approach. These papers assumed that we have two sources of information about one and the same variable and have shown that in psychophysical experiments people often show this kind of optimal integration and that the weights can be predicted from the variances [13, 14, 15, 4, 16]. However, in all these cases there is ample of evidence provided to the subjects that just one single variable is involved in the experiment. For example in [4] a stimulus is felt and seen at exactly the same position.

## 3   Combination of visual and auditory cues: uncertainty about the causal structure

Here we consider the range of experiments where people hear a tone and simultaneously see a visual stimulus that may or may not come from the same position. Subjects are asked to estimate which direction the tone is coming from and point to that direction – placing this experiment in the realm of sensorimotor integration.

Subjects are asked to estimate which direction the tone is coming from and do so with a motor response. To optimally estimate where the tone is coming from people need to infer the causal structure (Fig 1 C) and decide if they should assume a single cause or two causes. Based on this calculation they can proceed to estimate where the tone is coming from. The Schirillo group has extensively tested human behavior in such a situation [17, 18]. For different distances between the visual and the auditory stimulus they analyzed the strategies people use to estimate the position of the auditory stimuli (see figure 2). It has long been realized that integration of different cues should only occur if the cues have the same cause [9, 10, 8, 19].

## 3.1  Loss function and probabilistic model

To model this choice phenomenon we assume that the estimate should be as precise as possible and that this error function is minimized:

$$E(x_{estimated}) = \int p(x_{true}|cues)(x_{true} - x_{estimated})^2 dx_{true} \tag{4}$$

We assume that subjects have obtained a prior estimate $p_{same}$ of how likely it is that a visual and an auditory signal that appear near instantaneously have the same cause. In everyday life this will not be constant but depend on temporal delays, visual experience, context and many other factors. In the experiments we consider all these factors are held constant so we can use a constant $p_{same}$. We assume that positions are drawn from a Gaussian distribution with a width $\sigma_{pos}$.

## 3.2  Inference

The probability that the two signals are from the same source will only weakly depend on the spatial prior but mostly depend on the distance $\Delta_{av} = x_{aud} - x_{vis}$ between visually and auditory perceived positions. We thus obtain:

$$\frac{p(same|\Delta_{av})}{p(different|\Delta_{av})} = \frac{p_{same}p(\Delta_{av}|same)}{(1 - p_{same})p(\Delta_{av}|different)} \tag{5}$$

Using $p(same|\Delta_{av}) + p(different|\Delta_{av}) = 1$ we can readily calculate the probability $p(same|\Delta_{av})$ of the two signals coming from the same source.

Using Equation 4 we can then calculate the optimal solution which is:

$$\hat{x} = p(same|\Delta_{av})\hat{x}_{same} + (1 - p(same|\Delta_{av}))\hat{x}_{different} \tag{6}$$

We know the optimal estimates in the same case already from equation 3 and in the different case the optimal estimate exclusively relies on the auditory signal. We furthermore assume that the position sensed by the sensory system is a noisy version of $x_{observed} = \hat{x} + \epsilon$ where $\epsilon$ is drawn from a Gaussian with zero mean and a standard deviation of $\sigma_{motor}$. We are thus able to calculate the optimal estimate and the expected uncertainty given our assumptions.

## 3.3  Model parameter estimation

The prior $p_{same}$ characterizes how likely given the temporal delay and other experimental parameters it is a priori that two signals have the same source. As this characterizes a property of everyday life we can not readily estimate this parameter but instead fit it to the gain ($\alpha$) data. To compare the predictions of our model with the experimental data we need to know the values of the variables that characterize our model. Vision is much more precise than audition in such situations. We estimate the relevant uncertainties as follows. In both auditory and visual trials the noise will have two sources, motor noise and sensory noise. Even if people knew perfectly where a stimulus was coming from they would make small errors at pointing because their motor system is not perfect. We assume that visual only trials are dominated by motor noise, stemming from motor errors and memory errors and that the noise in the visual trials is essentially exclusively motor noise ($\sigma_{vis} = 0.01$). Choosing a smaller $\sigma_{vis}$ does not change the results to any meaningful degree. From figure 2 of the experiments by Hairston et al [17] where movements are made towards unimodally presented cues we obtain $\sigma_{motor} = 2.5$ deg and because variances are added linearly $\sigma_{aud} = \sqrt{8^2 - 2.5^2} = 7.6$ deg.

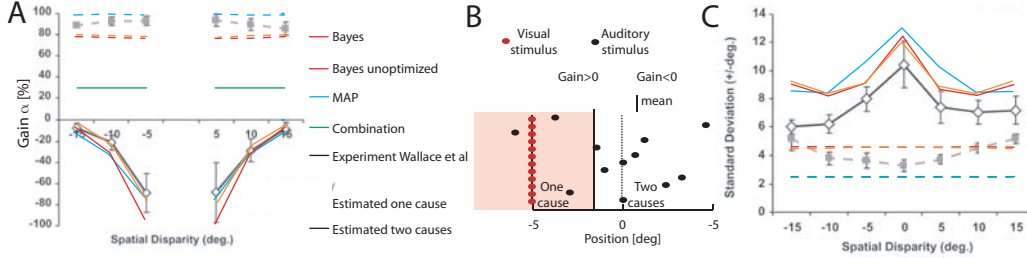

Figure 2: Uncertainty if one or two causes are relevant. Experimental data reprinted with permission from [18]. A) The gain, the relative weight of vision for the estimation of the position of the auditory signal is shown. It is plotted as a function of the spatial disparity, the distance between visual and auditory stimulus. A gain value of $\alpha = 100\%$ implies that subjects only use visual information. A negative $\alpha$ means that on average subjects point away from the visual stimulus. Different models of human behavior make different preditictions. B) A sketch explaining the finding of negative gains. The visual stimulus is always at -5 deg (solid line) and the auditory stimulus is always straight ahead at 0deg(dotted line). Visual perception is very low noise and the perceived position $x_{vis}$ is shown as red dots (each dot is one trial). Auditory perception is noisy and the perceived auditory position $x_{aud}$ is shown as black dots. In the white area where the subject perceive two causes, the average position of perceived auditory signals is further to the right. This explains the negative bias in reporting: when perceiving two causes, subjects are more likely to have heard a signal to the right. Those trials that are not unified thus exhibit a selection bias that confers the negative gain. C) The measured standard deviation of the human pointing behavior are shown as a function of the spatial disparity. The standard deviations predicted by the model are shown as well. Same colors as in A)

We want to remark that this estimation is only approximate because people can use priors and combine them with likelihoods and objective functions for making their estimates even in the unimodal case. We also want to emphasize that we in no way tried to tune these parameters to lead to better fits of the data. From the specifications of the experiments we know that the distribution of auditory sources has a width of 20deg relative to the fixation point and we assume that this width is known to the subjects from repeated trials.

### 3.4 Comparison of the model with the experimental data

Figure 2A shows a comparison between the gains ($\alpha$) measured in the experiment of [17] with the gains ($alpha$) predicted by the Bayesian model. $p_{same} = 0.57$ was fitted to the data. We assume that the model reports identical whenever one source is a posteriori more probable than two sources. The model predicts the counterintuitive finding that the trials where people inferred two causes exhibit negative gain. Figure 2B explains why negative gains are found. The model explains 99% of the variance of the gain with just one free parameter $p_{same}$. Very similar effects are found if we fix $p_{same}$ at 0.5 assuming that fusion and segregation are equally likely and this parameter free model still explains 98% of the variance. The simple full combination model (shown in green) that does not allow for two sources completely fails to predict any of these effects even when fitting all the standard deviations and thus explains 0% of the variance of the gains. The results clearly rule out a strategy in which all cues are always combined.

On some trials noise in the auditory signal will make it appear as if the auditory signal is very close to the visual signal. In this case the system will infer that both have the same source and part of the reported high gain for the fused cases will be because noise already perturbed the auditory signal towards the visual. However, on some trials the auditory signal will be randomly perturbed away from the visual signal. In this case the system will infer that very likely the two signals have different sources. Because both estimation of position and the estimation of identity are based on the same noisy signal the two processes are not independent of one another. This lack of independence is causing the difference between the fusion and the no-fusion case.

### 3.5 Maximum A Posteriori over causal structure

In the derivations above we assumed that people are fully Bayesian, in the sense that they consider both possible structures for cue-integration and integrate over them to arrive at an optimal decision. An alternative would be a Maximum A Posteriori (MAP) approach: people could first choose the appropriate structure  one source or two  and then use only that structure for subsequent decisions. Figure 2A shows that this model (we fitted $p_{same} = 0.72$) also well predicts the main effect and explains 98% of the variance of the gains. To test how the two models compare we looked at the standard deviations that had also been measured in the [17] experiment. The fully Bayesian model explains 65% of the variance of the standard deviation plot and the MAP model explains 0% of the variance of that plot. This difference is observed because the MAP model strongly underestimates the uncertainty in the single cause case and strongly overestimates the uncertainty in the dual cause case (Fig 2C). The Bayesian model on the other hand always considers that it could be wrong, leading to more variance in the single cause case and less in the dual cause case. Even the Bayesian system tends to predict overly large standard deviations in the case of two causes. This effect goes away if we assume that people underestimate the variance of the auditory source relative to the fixation spot (data not shown). A deeper analysis taking into account all the available data and its variance over subjects will be necessary to test if a MAP strategy can be ruled out. The present analysis may lead to an understanding of the inference algorithm used by the nervous system.

In summary, the problem of crossmodal integration is much more complicated than it seems as it necessitates inference of the causal structure. People still solve this complicated problem in a way that can be understood as being close to optimal.

## 4   Combination of visual and proprioceptive cues

Typical experiments in movement psychophysics where a virtual reality display is used to disturb the perceived position of the hand lead to an analogous problem. In these experiments subjects proprioceptively feel their hand somewhere, but they cannot see their hand; at the same time, they visually perceive a cursor somewhere. Subjects again can not be sure if the seen cursor position and the felt hand position are cues about the same variable (hand=cursor) or if each of them are independent and the experiment is just cheating them leading to the same causal structure inference problem described above. In this section we extend the model to also explain such sensorimotor integration.

We model the studies by Sober and Sabes [5, 6] that inspired this work. In these experiments one hand is moving a cursor to either the position of a visually displayed (v) target or the position of the other hand (p). People need to estimate two distinct variables: (1) the direction in which they are to move their arm, a visually perceived variable, the so-called movement vector (MV) and (2) a proprioceptively perceived variable, the configuration of their joints (J). Subjects obtain visual information about the position of the cursor and they obtain proprioceptive information from feeling the position of their hand.

Traditionally it would have been assumed that the seen cursor position and the proprioceptively felt hand position are cues caused by one single variable, the hand. As a result, the position of the cursor uniquely defines the configuration of the joints and vice versa. As in the cue combination case above there should not be full cue combination but instead each variable (MV) and (J) should be estimated separately. In this experiment a situation is produced where the visual position of a cursor is different from the actual position of the right hand. Subjects are then asked to move their hand towards targets that are in 8 concentric directions. The estimate of the movement vector affects movements direction in a way that is specific to the target direction. The estimate of the joint configuration affects movement direction irrespective of the target direction. The experimental studies then report the gain $\alpha$, the linear weight $\alpha$ of vision on the estimate of (MV) and (J) in both the visual and the proprioceptive target conditions(figure 3A and B). If people only inferred about one common cause then the weight of vision should always be the same, indicating that more than just one cause is assumed by the subjects.

## 4.1  Coordinate systems

The probabilistic model that we use is identical to the model introduced above with one exception. In the sensorimotor integration case there is uncertainty about the alignment of the two coordinate systems. For example if we hold an arm under a table and where asked to show where the other arm is under the table we would have significant alignment errors. When using information from one coordinate system for an estimation in a different coordinate system there is uncertainty about the alignment of the coordinate systems. This means that when we use visual information to estimate the position of a joint in joint space our visual system appears to be more noisy and vice versa. As we are only interested in estimates along one dimension and can model the uncertainty about the alignment of the coordinate systems as a one dimensional Gaussian with width $\sigma_{trans}$. When using information from one modality for estimations of a variable in the other coordinate system we need to use $\sigma^2_{effective} = \sigma^2_{modality} + \sigma^2_{trans}$.

The two target conditions in the experiments, moving the cursor to a visual target (v) and moving the cursor to the position of the other hand (p) produce two different estimation problems. When we try to move a cursor to a visually displayed target we must compute MV in visual space. If to the contrary we try to move a cursor with one hand to the position of the other hand then we must calculate MV in joint space. Loss functions and therefore necessary estimates are thus defined in different spaces. Altogether people are faced with 4 problems, they have to estimate (MV) and (J) in both the visual (v) condition and the proprioceptive (p) condition.

## 4.2  Probabilistic model

As above we assume that visual and proprioceptive uncertainty lead to probability distributions in the respective space that are characterized by Gaussians of width $\sigma_{vis}$ and $\sigma_{prop}$. These variables are now defined in terms of position not in terms of direction. Subjects are not asked if they experience one or two causes. Under these circumstances it is only important how likely on average people find that the two percepts are unified ($p_{unified} = p_{same}p(\Delta_{pv}|same)$). We assume that when moving the cursor to a visual target the average squared deviation of the cursor and the target in visual space is minimized. We assume that when moving the cursor to a proprioceptive target the average squared deviation of the cursor and the target in proprioceptive space is minimized. Apart from this difference the whole derivation of the equations is identical to the one above for the auditory visual integration. However, the results are not analyzed conditional on the inference of one or two causes but averaged over these.

## 4.3  Tool use

Above we assumed that cursor and hand either have the same cause (the position of the hand, or different causes and are therefore unrelated. Another way of thinking about the Sober and Sabes experiments could be in terms of tool use. The cursor could be seen as a tool that is seen displaced relative to our hand. The tip of the tool will move with our hand. As tools are typically short the probability is largest that the tip of a tool is at the position of the hand and this probability will decay with increasing distance between the hand and the position of the tool. The distance between the tip of the tool and the hand is thus another random variable that is assumed to be Gaussian with width $\sigma_{tool}$ (see fig. 3E). The minimal end point error solutions of this are:

$$\alpha_{MV,v} = (\sigma^2_{prop} + \sigma^2_{trans} + \sigma^2_{tool})/(\sigma^2_{prop} + \sigma^2_{vis} + \sigma^2_{trans} + \sigma^2_{tool}) \quad (7)$$

$$\alpha_{J,v} = (\sigma^2_{prop} + \sigma^2_{trans})/(\sigma^2_{prop} + \sigma^2_{vis} + \sigma^2_{trans} + \sigma^2_{tool}) \quad (8)$$

$$\alpha_{MV,p} = (\sigma^2_{prop} + \sigma^2_{tool})/(\sigma^2_{prop} + \sigma^2_{vis} + \sigma^2_{trans} + \sigma^2_{tool}) \quad (9)$$

$$\alpha_{J,p} = (\sigma^2_{prop})/(\sigma^2_{prop} + \sigma^2_{vis} + \sigma^2_{trans} + \sigma^2_{tool}) \quad (10)$$

We are thus able to predict the weights that people should use if they assume a causal relationship deriving from tool use.

## 4.4  Comparison of the model with the data

We add to the Bayesian model introduced above a part for modeling the uncertainty about the alignment of the coordinate systems, and compare the results from this modified model with the data. The

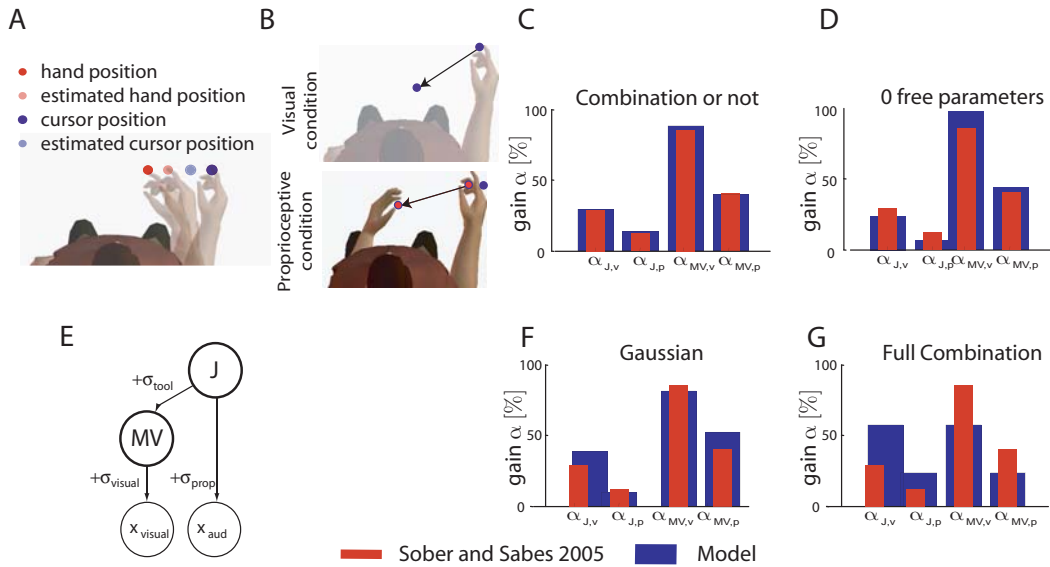

Figure 3: Cue combination in motor control, experiments from [6] A) The estimated quantities. B) The two experimental conditions. C)The predictions from the model. D)The predictions obtained when using the estimate of a specialist. E) The tool use model. The cursor will be close to the position of the hand. F)The predictions from a tool use model. G)The predictions from a full combination model.

model has several parameters, important the uncertainties of proprioception and of the coordinate transformation compared to the visual uncertainty. Another parameter is the probability of unification. All parameters are fit to the data. The model explains the data that have a standard deviation of 0.32 with a standard deviation of only 0.08 (Figure 3C). Fitting 3 parameters to 4 data points can be seen as some major overfitting. To avoid overfitting we guessed $p_{unified} = 0.5$ and asked one of our colleagues,, Daniel Wolpert, for estimates. He estimated $\sigma_{vis} = 1cm$,$\sigma_{prop} = 3cm$,$\sigma_{trans} = 5cm$. With these values we explain the data with a standard deviation of 0.13 capturing all main effects(Figure 3D). Another experimental modification in [6] deserves mentioning. The image of an arm is rendered on top of the cursor position. The experiment finds that this has the effect that people rely much more on vision for estimating their joint configuration. In our interpretation, the rendering of the arm makes the probability much higher that actually the position of the visual display is the position of the hand and $p_{unified}$ would be much higher.

### 4.5    Analysis if subjects view a cursor as a tool

Another possible model that seemed very likely to us was assuming that the cursor should appear somewhere close to the hand modeling the cursor hand relationship as another Gaussian variable (Fig 3E). We fit the 3 parameters of this model, the uncertainty of proprioception and the coordinate transformation relative to the visual uncertainty as well as the width of the Gaussian describing the tool. Figure 3F shows that this model too can fit the main results of the experiment. With a standard deviation of the residual of 0.14 however it does worse than the parameter free model above. If we take the values given by Daniel Wolpert (see above) and fit the value of $\sigma_{tool}$ we obtain a standard deviation of 0.28. This model of tool use seems to thus be doing poorer than the model we introduced earlier.

Sober and Sabes [5, 6] explain the finding that two variables are estimated by the finding that cortex exhibits two important streams of information processing, one for visual processing and the other for motor tasks [20]. The model we present here gives a reason for the estimation of distinct variables. If people see a cursor close to their hand they do not assume that they actually see their hand. The models that we introduced can be understood as special instantiations of a model where the cursor position relative to the hand is drawn from a general probability distribution.

# 5 Discussion

An impressive range of recent studies show that people do not just estimate one variable in situations of cue combination [5, 6, 17, 18]. Here we have shown that the statistical problem that people solve in such situations involves an inference about the causal structure. People have uncertainty about the identity and number of relevant variables. The problem faced by the nervous system is similar to cognitive problems that occur in the context of causal induction. Many experiments show that people and in particular infants interpret events in terms of cause and effect [11, 21, 22]. The results presented here show that sensorimotor integration exhibits some of the factors that make human cognition difficult. Carefully studying and analyzing seemingly simple problems such as cue combination may provide a fascinating way of studying the human cognitive system in a quantitative fashion.

# References

[1] Q. Haijiang, J. A. Saunders, R. W. Stone, and B. T. Backus. Demonstration of cue recruitment: change in visual appearance by means of pavlovian conditioning. *Proc Natl Acad Sci U S A*, 103(2):483–8, 2006. 0027-8424 (Print) Journal Article.

[2] J. W. Krakauer, M. F. Ghilardi, and C. Ghez. Independent learning of internal models for kinematic and dynamic control of reaching. *Nat Neurosci*, 2(11):1026–31, 1999.

[3] R. Shadmehr and F. A. Mussa-Ivaldi. Adaptive representation of dynamics during learning of a motor task. *J Neurosci*, 14(5 Pt 2):3208–24, 1994.

[4] M. O. Ernst and M. S. Banks. Humans integrate visual and haptic information in a statistically optimal fashion. *Nature*, 415(6870):429–33, 2002.

[5] S. J. Sober and P. N. Sabes. Multisensory integration during motor planning. *J Neurosci*, 23(18):6982–92, 2003.

[6] S. J. Sober and P. N. Sabes. Flexible strategies for sensory integration during motor planning. *Nat Neurosci*, 8(4):490–7, 2005.

[7] K. P. Koerding and D. M. Wolpert. Bayesian integration in sensorimotor learning. *Nature*, 427(6971):244–7, 2004.

[8] L. Shams, W. J. Ma, and U. Beierholm. Sound-induced flash illusion as an optimal percept. *Neuroreport*, 16(17):1923–7, 2005.

[9] E Hirsch. *The concept of Identity*. Oxford University Press, Oxford, 1982.

[10] A. Leslie, F. Xu, P. Tremoulet, and B. Scholl. Indexing and the object concept: "what" and "where" in infancy. *Trends in Cognitive Sciences*, 2:10–18, 1998.

[11] A. Gopnik, C. Glymour, D. M. Sobel, L. E. Schulz, T. Kushnir, and D. Danks. A theory of causal learning in children: causal maps and bayes nets. *Psychol Rev*, 111(1):3–32, 2004.

[12] T. L. Griffiths and J. B. Tenenbaum. From mere coincidences to meaningful discoveries. *Cognition*, 2006. 0010-0277 (Print) Journal article.

[13] Z. Ghahramani. *Computational and psychophysics of sensorimotor integration*. PhD thesis, Massachusetts Institute of Technology, 1995.

[14] R. A. Jacobs. Optimal integration of texture and motion cues to depth. *Vision Res*, 39(21):3621–9, 1999.

[15] R. J. van Beers, A. C. Sittig, and J. J. Gon. Integration of proprioceptive and visual position-information: An experimentally supported model. *J Neurophysiol*, 81(3):1355–64, 1999.

[16] D. Alais and D. Burr. The ventriloquist effect results from near-optimal bimodal integration. *Curr Biol*, 14(3):257–62, 2004.

[17] W. D. Hairston, M. T. Wallace, J. W. Vaughan, B. E. Stein, J. L. Norris, and J. A. Schirillo. Visual localization ability influences cross-modal bias. *J Cogn Neurosci*, 15(1):20–9, 2003.

[18] M. T. Wallace, G. E. Roberson, W. D. Hairston, B. E. Stein, J. W. Vaughan, and J. A. Schirillo. Unifying multisensory signals across time and space. *Exp Brain Res*, 158(2):252–8, 2004.

[19] Shams L Beierholm U, Quartz S. Bayesian inference as a unifying model of auditory-visual integration and segregation. In *Proceedings of the society of neuroscience*, 2005.

[20] M. A. Goodale, G. Kroliczak, and D. A. Westwood. Dual routes to action: contributions of the dorsal and ventral streams to adaptive behavior. *Prog Brain Res*, 149:269–83, 2005.

[21] R. Saxe, J. B. Tenenbaum, and S. Carey. Secret agents: inferences about hidden causes by 10- and 12-month-old infants. *Psychol Sci*, 16(12):995–1001, 2005.

[22] T. L. Griffiths and J. B. Tenenbaum. Structure and strength in causal induction. *Cognit Psychol*, 51(4):334–84, 2005. 0010-0285 (Print) Journal Article.
